# Learning Continuous Attractors in Recurrent Networks

**H. Sebastian Seung**
Bell Labs, Lucent Technologies
Murray Hill, NJ 07974
seung@bell-labs.com

## Abstract

One approach to invariant object recognition employs a recurrent neural network as an associative memory. In the standard depiction of the network's state space, memories of objects are stored as attractive fixed points of the dynamics. I argue for a modification of this picture: if an object has a continuous family of instantiations, it should be represented by a continuous attractor. This idea is illustrated with a network that learns to complete patterns. To perform the task of filling in missing information, the network develops a continuous attractor that models the manifold from which the patterns are drawn. From a statistical viewpoint, the pattern completion task allows a formulation of unsupervised learning in terms of regression rather than density estimation.

A classic approach to invariant object recognition is to use a recurrent neural network as an associative memory[1]. In spite of the intuitive appeal and biological plausibility of this approach, it has largely been abandoned in practical applications. This paper introduces two new concepts that could help resurrect it: object representation by continuous attractors, and learning attractors by pattern completion.

In most models of associative memory, memories are stored as attractive fixed points at discrete locations in state space[1]. Discrete attractors may not be appropriate for patterns with continuous variability, like the images of a three-dimensional object from different viewpoints. When the instantiations of an object lie on a continuous *pattern manifold*, it is more appropriate to represent objects by attractive manifolds of fixed points, or continuous attractors.

To make this idea practical, it is important to find methods for learning attractors from examples. A naive method is to train the network to retain examples in short-term memory. This method is deficient because it does not prevent the network from storing spurious fixed points that are unrelated to the examples. A superior method is to train the network to restore examples that have been corrupted, so that it learns to complete patterns by filling in missing information.

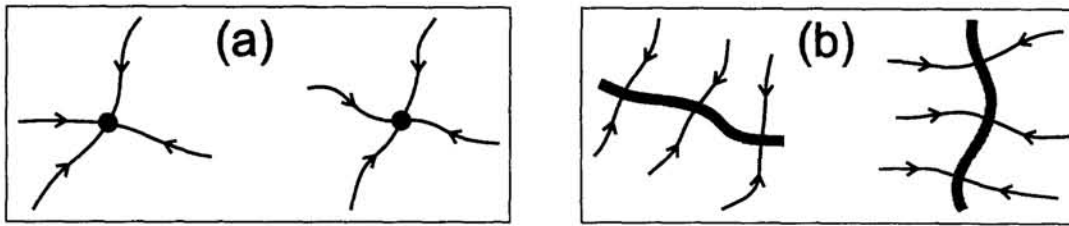

Figure 1: Representing objects by dynamical attractors. (a) Discrete attractors. (b) Continuous attractors.

Learning by pattern completion can be understood from both dynamical and statistical perspectives. Since the completion task requires a large basin of attraction around each memory, spurious fixed points are suppressed. The completion task also leads to a formulation of unsupervised learning as the regression problem of estimating functional dependences between variables in the sensory input.

Density estimation, rather than regression, is the dominant formulation of unsupervised learning in stochastic neural networks like the Boltzmann machine[2]. Density estimation has the virtue of suppressing spurious fixed points automatically, but it also has the serious drawback of being intractable for many network architectures. Regression is a more tractable, but nonetheless powerful, alternative to density estimation.

In a number of recent neurobiological models, continuous attractors have been used to represent continuous quantities like eye position [3], direction of reaching[4], head direction[5], and orientation of a visual stimulus[6]. Along with these models, the present work is part of a new paradigm for neural computation based on continuous attractors.

# 1   DISCRETE VERSUS CONTINUOUS ATTRACTORS

Figure 1 depicts two ways of representing objects as attractors of a recurrent neural network dynamics. The standard way is to represent each object by an attractive fixed point[1], as in Figure 1a. Recall of a memory is triggered by a sensory input, which sets the initial conditions. The network dynamics converges to a fixed point, thus retrieving a memory. If different instantiations of one object lie in the same basin of attraction, they all trigger retrieval of the same memory, resulting in the many-to-one map required for invariant recognition.

In Figure 1b, each object is represented by a continuous manifold of fixed points. A one-dimensional manifold is shown, but generally the attractor should be multidimensional, and is parametrized by the instantiation or pose parameters of the object. For example, in visual object recognition, the coordinates would include the viewpoint from which the object is seen.

The reader should be cautioned that the term "continuous attractor" is an idealization and should not be taken too literally. In real networks, a continuous attractor is only approximated by a manifold in state space along which drift is very slow. This is illustrated by a simple example, a descent dynamics on a trough-shaped energy landscape[3]. If the bottom of the trough is perfectly level, it is a line of fixed points and an ideal continuous attractor of the dynamics. However, any slight imperfections cause slow drift along the line. This sort of approximate continuous attractor is what is found in real networks, including those trained by the learning

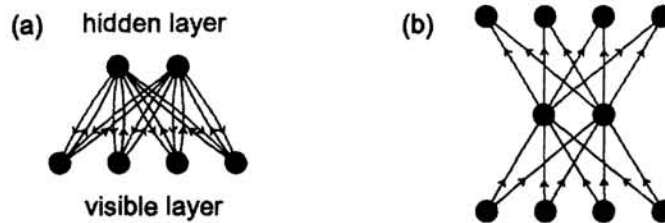

Figure 2: (a) Recurrent network. (b) Feedforward autoencoder.

algorithms to be discussed below.

## 2 DYNAMICS OF MEMORY RETRIEVAL

The preceding discussion has motivated the idea of representing pattern manifolds by continuous attractors. This idea will be further developed with the simple network shown in Figure 2a, which consists of a visible layer $x_1 \in R^{n_1}$ and a hidden layer $x_2 \in R^{n_2}$. The architecture is recurrent, containing both bottom-up connections (the $n_2 \times n_1$ matrix $W_{21}$) and top-down connections (the $n_1 \times n_2$ matrix $W_{12}$). The vectors $b_1$ and $b_2$ represent the biases of the neurons. The neurons have a rectification nonlinearity $[x]^+ = \max\{x, 0\}$, which acts on vectors component by component.

There are many variants of recurrent network dynamics: a convenient choice is the following discrete-time version, in which updates of the hidden and visible layers alternate in time. After the visible layer is initialized with the input vector $x_1(0)$, the dynamics evolves as

$$
\begin{aligned}
x_2(t) &= [b_2 + W_{21}x_1(t-1)]^+ , \\
x_1(t) &= [b_1 + W_{12}x_2(t)]^+ .
\end{aligned}
\tag{1}
$$

If memories are stored as attractors, iteration of this dynamics can be regarded as memory retrieval.

Activity circulates around the feedback loop between the two layers. One iteration of this loop is the map $x_1(t-1) \to x_2(t) \to x_1(t)$. This single iteration is equivalent to the feedforward architecture of Figure 2b. In the case where the hidden layer is smaller than the visible layers, this architecture is known as an autoencoder network[7]. Therefore the recurrent network dynamics (1) is equivalent to repeated iterations of the feedforward autoencoder. This is just the standard trick of unfolding the dynamics of a recurrent network in time, to yield an equivalent feedforward network with many layers[7]. Because of the close relationship between the recurrent network of Figure 2a and the autoencoder of Figure 2b, it should not be surprising that learning algorithms for these two networks are also related, as will be explained below.

## 3 LEARNING TO RETAIN PATTERNS

Little trace of an arbitrary input vector $x_1(0)$ remains after a few time steps of the dynamics (1). However, the network can retain some input vectors in short-term memory as "reverberating" patterns of activity. These correspond to fixed points of the dynamics (1); they are patterns that do not change as activity circulates around the feedback loop.

This suggests a formulation of learning as the optimization of the network's ability to retain examples in short-term memory. Then a suitable cost function is the squared difference $|x_1(T) - x_1(0)|^2$ between the example pattern $x_1(0)$ and the network's short-term memory $x_1(T)$ of it after $T$ time steps. Gradient descent on this cost function can be done via backpropagation through time[7].

If the network is trained with patterns drawn from a continuous family, then it can learn to perform the short-term memory task by developing a continuous attractor that lies near the examples it is trained on. When the hidden layer is smaller than the visible layer, the dimensionality of the attractor is limited by the size of the hidden layer.

For the case of a single time step $(T = 1)$, training the recurrent network of Figure 2a to retain patterns is equivalent to training the autoencoder of Figure 2b by minimizing the squared difference between its input and output layers, averaged over the examples[8]. From the information theoretic perspective, the small hidden layer in Figure 2b acts as a bottleneck between the input and output layers, forcing the autoencoder to learn an efficient encoding of the input.

For the special case of a linear network, the nature of the learned encoding is understood completely. Then the input and output vectors are related by a simple matrix multiplication. The rank of the matrix is equal to the number of hidden units. The average distortion is minimized when this matrix becomes a projection operator onto the subspace spanned by the principal components of the examples[9].

From the dynamical perspective, the principal subspace is a continuous attractor of the dynamics (1). The linear network dynamics converges to this attractor in a single iteration, starting from any initial condition. Therefore we can interpret principal component analysis and its variants as methods of learning continuous attractors[10].

## 4 LEARNING TO COMPLETE PATTERNS

Learning to retain patterns in short-term memory only works properly for architectures with a small hidden layer. The problem with a large hidden layer is evident when the hidden and visible layers are the same size, and the neurons are linear. Then the cost function for learning can be minimized by setting the weight matrices equal to the identity, $W_{21} = W_{12} = I$. For this trivial minimum, every input vector is a fixed point of the recurrent network (Figure 2a), and the equivalent feedforward network (Figure 2b) exactly realizes the identity map. Clearly these networks have not learned anything.

Therefore in the case of a large hidden layer, learning to retain patterns is inadequate. Without the bottleneck in the architecture, there is no pressure on the feedforward network to learn an efficient encoding. Without constraints on the dimension of the attractor, the recurrent network develops spurious fixed points that have nothing to do with the examples.

These problems can be solved by a different formulation of learning based on the task of pattern completion. In the completion task of Figure 3a, the network is initialized with a corrupted version of an example. Learning is done by minimizing the completion error, which is the squared difference $|x_1(T) - d|^2$ between the uncorrupted pattern $d$ and the final visible vector $x_1(T)$. Gradient descent on completion error can be done with backpropagation through time[11].

This new formulation of learning eliminates the trivial identity map solution men-

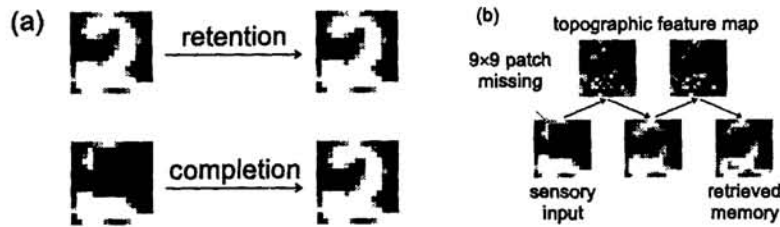

Figure 3: (a) Pattern retention versus completion. (b) Dynamics of pattern completion.

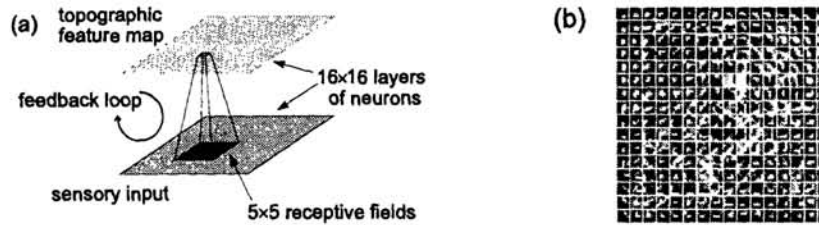

Figure 4: (a) Locally connected architecture. (b) Receptive fields of hidden neurons.

tioned above: while the identity network can retain any example, it cannot restore corrupted examples to their pristine form. The completion task forces the network to enlarge the basins of attraction of the stored memories, which suppresses spurious fixed points. It also forces the network to learn associations between variables in the sensory input.

## 5  LOCALLY CONNECTED ARCHITECTURE

Experiments were conducted with images of handwritten digits from the USPS database described in [12]. The example images were 16 × 16, with a gray scale ranging from 0 to 1. The network was trained on a specific digit class, with the goal of learning a single pattern manifold. Both the network architecture and the nature of the completion task were chosen to suit the topographic structure present in visual images.

The network architecture was given a topographic organization by constraining the synaptic connectivity to be local, as shown in Figure 4a. Both the visible and hidden layers of the network were 16 × 16. The visible layer represented an image, while the hidden layer was a topographic feature map. Each neuron had 5 × 5 receptive and projective fields, except for neurons near the edges, which had more restricted connectivity.

In the pattern completion task, example images were corrupted by zeroing the pixels inside a 9 × 9 patch chosen at a random location, as shown in Figure 3a. The location of the patch was randomized for each presentation of an example. The size of the patch was a substantial fraction of the 16 × 16 image, and much larger than the 5 × 5 receptive field size. This method of corrupting the examples gave the completion task a topographic nature, because it involved a set of spatially contiguous pixels. This topographic nature would have been lacking if the examples had been corrupted by, for example, the addition of spatially uncorrelated noise.

Figure 3b illustrates the dynamics of pattern completion performed by a network

trained on examples of the digit class "two." The network is initialized with a corrupted example of a "two." After the first iteration of the dynamics, the image is partially restored. The second iteration leads to superior restoration, with further sharpening of the image. The "filling in" phenomenon is also evident in the hidden layer.

The network was first trained on a retrieval dynamics of one iteration. The resulting biases and synaptic weights were then used as initial conditions for training on a retrieval dynamics of two iterations. The hidden layer developed into a topographic feature map suitable for representing images of the digit "two." Figure 4b depicts the bottom-up receptive fields of the 256 hidden neurons. The top-down projective fields of these neurons were similar, but are not shown.

This feature map is distinct from others[13] because of its use of top-down and bottom-up connections in a feedback loop. The bottom-up connections analyze images into their constituent features, while the top-down connections synthesize images by composing features. The features in the top-down connections can be regarded as a "vocabulary" for synthesis of images. Since not all combinations of features are proper patterns, there must be some "grammatical" constraints on their combination. The network's ability to complete patterns suggests that some of these constraints are embedded in the dynamical equations of the network. Therefore the relaxation dynamics (1) can be regarded as a process of massively parallel constraint satisfaction.

# 6   CONCLUSION

I have argued that continuous attractors are a natural representation for pattern manifolds. One method of learning attractors is to train the network to retain examples in short-term memory. This method is equivalent to autoencoder learning, and does not work if the number of hidden units is large. A better method is to train the network to complete patterns. For a locally connected network, this method was demonstrated to learn a topographic feature map. The trained network is able to complete patterns, indicating that syntactic constraints on the combination of features are embedded in the network dynamics.

Empirical evidence that the network has indeed learned a continuous attractor is obtained by local linearization of the network (1). The linearized dynamics has many eigenvalues close to unity, indicating the existence of an approximate continuous attractor. Learning with an increased number of iterations in the retrieval dynamics should improve the quality of the approximation.

There is only one aspect of the learning algorithm that is specifically tailored for *continuous* attractors. This aspect is the limitation of the retrieval dynamics (1) to a few iterations, rather than iterating it all the way to a true fixed point. As mentioned earlier, a continuous attractor is only an idealization; in a real network it does not consist of true fixed points, but is just a manifold to which relaxation is fast and along which drift is slow. Adjusting the shape of this manifold is the goal of learning; the exact locations of the true fixed points are not relevant.

The use of a fast retrieval dynamics removes one long-standing objection to attractor neural networks, which is that true convergence to a fixed point takes too long. If all that is desired is fast relaxation to an approximate continuous attractor, attractor neural networks are not much slower than feedforward networks.

In the experiments discussed here, learning was done with backpropagation through time. Contrastive Hebbian learning[14] is a simpler alternative. Part of the image

is held clamped, the missing values are filled in by convergence to a fixed point, and an anti-Hebbian update is made. Then the missing values are clamped at their correct values, the network converges to a new fixed point, and a Hebbian update is made. This procedure has the disadvantage of requiring true convergence to a fixed point, which can take many iterations. It also requires symmetric connections, which may be a representational handicap.

This paper addressed only the learning of a single attractor to represent a single pattern manifold. The problem of learning multiple attractors to represent multiple pattern classes will be discussed elsewhere, along with the extension to network architectures with many layers.

**Acknowledgments**   This work was supported by Bell Laboratories. I thank J. J. Hopfield, D. D. Lee, L. K. Saul, N. D. Socci, H. Sompolinsky, and D. W. Tank for helpful discussions.

# References

[1] J. J. Hopfield. Neural networks and physical systems with emergent collective computational abilities. *Proc. Nat. Acad. Sci. USA*, 79:2554–2558, 1982.

[2] D. H. Ackley, G. E. Hinton, and T. J. Sejnowski. A learning algorithm for Boltzmann machines. *Cognitive Science*, 9:147–169, 1985.

[3] H. S. Seung. How the brain keeps the eyes still. *Proc. Natl. Acad. Sci. USA*, 93:13339–13344, 1996.

[4] A. P. Georgopoulos, M. Taira, and A. Lukashin. Cognitive neurophysiology of the motor cortex. *Science*, 260:47–52, 1993.

[5] K. Zhang. Representation of spatial orientation by the intrinsic dynamics of the head-direction cell ensemble: a theory. *J. Neurosci.*, 16:2112–2126, 1996.

[6] R. Ben-Yishai, R. L. Bar-Or, and H. Sompolinsky. Theory of orientation tuning in visual cortex. *Proc. Nat. Acad. Sci. USA*, 92:3844–3848, 1995.

[7] D.E. Rumelhart, G.E. Hinton, and R.J. Williams. Learning internal representations by error propagation. In D.E. Rumelhart and J.L. McClelland, editors, *Parallel Distributed Processing*, volume 1, chapter 8, pages 318–362. MIT Press, Cambridge, 1986.

[8] G. W. Cottrell, P. Munro, and D. Zipser. Image compression by back propagation: an example of extensional programming. In N. E. Sharkey, editor, *Models of cognition: a review of cognitive science*. Ablex, Norwood, NJ, 1989.

[9] P. Baldi and K. Hornik. Neural networks and principal component analysis: Learning from examples without local minima. *Neural Networks*, 2:53–58, 1989.

[10] H. S. Seung. Pattern analysis and synthesis in attractor neural networks. In K.-Y. M. Wong, I. King, and D.-Y. Yeung, editors, *Theoretical Aspects of Neural Computation: A Multidisciplinary Perspective*, Singapore, 1997. Springer-Verlag.

[11] F.-S. Tsung and G. W. Cottrell. Phase-space learning. *Adv. Neural Info. Proc. Syst.*, 7:481–488, 1995.

[12] Y. LeCun et al. Learning algorithms for classification: a comparison on handwritten digit recognition. In J.-H. Oh, C. Kwon, and S. Cho, editors, *Neural networks: the statistical mechanics perspective*, pages 261–276, Singapore, 1995. World Scientific.

[13] T. Kohonen. The self-organizing map. *Proc. IEEE*, 78:1464–1480, 1990.

[14] J. J. Hopfield, D. I. Feinstein, and R. G. Palmer. "Unlearning" has a stabilizing effect in collective memories. *Nature*, 304:158–159, 1983.